# Ensemble weighted kernel estimators for multivariate entropy estimation

**Kumar Sricharan, Alfred O. Hero III**
Department of EECS
University of Michigan
Ann Arbor, MI 48104
{kksreddy,hero}@umich.edu

## Abstract

The problem of estimation of entropy functionals of probability densities has received much attention in the information theory, machine learning and statistics communities. Kernel density plug-in estimators are simple, easy to implement and widely used for estimation of entropy. However, for large feature dimension $d$, kernel plug-in estimators suffer from the curse of dimensionality: the MSE rate of convergence is glacially slow - of order $O(T^{-\gamma/d})$, where $T$ is the number of samples, and $\gamma > 0$ is a rate parameter. In this paper, it is shown that for sufficiently smooth densities, an ensemble of kernel plug-in estimators can be combined via a weighted convex combination, such that the resulting weighted estimator has a superior parametric MSE rate of convergence of order $O(T^{-1})$. Furthermore, it is shown that these optimal weights can be determined by solving a convex optimization problem which does not require training data or knowledge of the underlying density, and therefore can be performed offline. This novel result is remarkable in that, while each of the individual kernel plug-in estimators belonging to the ensemble suffer from the curse of dimensionality, by appropriate ensemble averaging we can achieve parametric convergence rates.

## 1    Introduction

Non-linear entropy functionals of a multivariate density $f$ of the form $\int g(f(x), x)f(x)dx$ arise in applications including machine learning, signal processing, mathematical statistics, and statistical communication theory. Important examples of such functionals include Shannon and Rényi entropy. Entropy based applications include image registration and texture classification, ICA, anomaly detection, data and image compression, testing of statistical models and parameter estimation. For details and other applications, see, for example, Beirlant *et al.* [2] and Leonenko *et al.* [18]. In these applications, the functional of interest must be estimated empirically from sample realizations of the underlying densities. Several estimators of entropy measures have been proposed for general multivariate densities $f$. These include consistent estimators based on histograms [10, 2], kernel density plug-in estimators, entropic graphs [5, 20], gap estimators [24] and nearest neighbor distances [8, 18, 19].

Kernel density plug-in estimators [1, 6, 11, 15, 12] are simple, easy to implement, computationally fast and therefore widely used for estimation of entropy [2, 23, 14, 4, 13]. However, these estimators suffer from mean squared error (MSE) rates which typically grow with feature dimension $d$ as $O(T^{-\gamma/d})$, where $T$ is the number of samples and $\gamma$ is a positive rate parameter.

In this paper, we propose a novel weighted ensemble kernel density plug-in estimator of entropy $\hat{\mathbf{G}}_w$, that achieves parametric MSE rates of $O(T^{-1})$ when the feature density is smooth. The estimator is constructed as a weighted convex combination $\hat{\mathbf{G}}_w = \sum_{l \in \bar{l}} w(l) \hat{\mathbf{G}}_{k(l)}$ of individual kernel density plug-in estimators $\hat{\mathbf{G}}_{k(l)}$ wrt the weights $\{w(l); l \in \bar{l}\}$. Here, $\bar{l}$ is a vector of indices $\{l_1, .., l_L\}$ and $k(l) = l\sqrt{T/2}$ is proportional to the the volume of the kernel bins used in evaluating $\hat{\mathbf{G}}_{k(l)}$. The individual kernel estimators $\hat{\mathbf{G}}_{k(l)}$ are similar to the data-split kernel estimator of Györfi and van der Muelen [11], and have slow MSE rates of convergence of order $O(T^{-1/1+d})$. Please refer to Section 2 for the exact definition of $\hat{\mathbf{G}}_{k(l)}$.

The principal result presented in this paper is as follows. It is shown that the weights $\{w(l); l \in \bar{l}\}$ can be chosen so as to significantly improve the rate of MSE convergence of the weighted estimator $\hat{\mathbf{G}}_w$. In fact our ensemble averaging method can improve MSE convergence of $\hat{\mathbf{G}}_w$ to the parametric rate $O(T^{-1})$. These optimal weights can be determined by solving a convex optimization problem. Furthermore, this optimization problem does not involve any density-dependent parameters and can therefore be performed offline.

## 1.1 Related work

Ensemble based methods have been previously proposed in the context of classification. For example, in both boosting [21] and multiple kernel learning [16] algorithms, lower complexity weak learners are combined to produce classifiers with higher accuracy. Our work differs from these methods in several ways. First and foremost, our proposed method performs estimation rather than classification. An important consequence of this is that the weights we use are *data independent*, while the weights in boosting and multiple kernel learning must be estimated from training data since they depend on the unknown distribution.

Birge and Massart [3] show that for density $f$ in a Holder smoothness class with $s$ derivatives, the minimax MSE rate for estimation of a smooth functional is $T^{-2\gamma}$, where $\gamma = \min\{1/2, 4s/(4s + d)\}$. This means that for $s > 4/d$, parametric rates are achievable. The kernel estimators proposed in this paper require higher order smoothness conditions on the density, i. e. the density must be $s > d$ times differentiable. While there exist other estimators [17, 7] that achieve parametric MSE rates of $O(1/T)$ when $s > 4/d$, these estimators are more difficult to implement than kernel density estimators, which are a staple of many toolboxes in machine learning, pattern recognition, and statistics. The proposed ensemble weighted estimator is a simple weighted combination of off-the-shelf kernel density estimators.

## 1.2 Organization

The reminder of the paper is organized as follows. We formally describe the kernel plug-in entropy estimators for entropy estimation in Section 2 and discuss the MSE convergence properties of these estimators. In particular, we establish that these estimators have MSE rate which decays as $O(T^{-1/1+d})$. Next, we propose the weighted ensemble of kernel entropy estimators in Section 3. Subsequently, we provide an MSE-optimal set of weights as the solution to a convex optimization(3.4) and show that the resultant optimally weighted estimator has a MSE of $O(T^{-1})$. We present simulation results in Section 4 that illustrate the superior performance of this ensemble entropy estimator in the context of (i) estimation of the Panter-Dite distortion-rate factor [9] and (ii) testing the probability distribution of a random sample. We conclude the paper in Section 5.

## Notation

We will use bold face type to indicate random variables and random vectors and regular type face for constants. We denote the expectation operator by the symbol $\mathbb{E}$, the variance operator as $\mathbb{V}[\mathbf{X}] = \mathbb{E}[(\mathbf{X} - \mathbb{E}[\mathbf{X}])^2]$, and the bias of an estimator by $\mathbb{B}$.

## 2 Entropy estimation

This paper focuses on the estimation of general non-linear functionals $G(f)$ of $d$-dimensional multivariate densities $f$ with known support $\mathcal{S} = [a, b]^d$, where $G(f)$ has the form

$$G(f) = \int g(f(x), x) f(x) d\mu(x), \tag{2.1}$$

for some smooth function $g(f, x)$. Let $\mathcal{B}$ denote the boundary of $\mathcal{S}$. Here, $\mu$ denotes the Lebesgue measure and $\mathbb{E}$ denotes statistical expectation with respect to the density $f$. Assume that $T = N + M$ i.i.d realizations of feature vectors $\{\mathbf{X}_1, \ldots, \mathbf{X}_N, \mathbf{X}_{N+1}, \ldots, \mathbf{X}_{N+M}\}$ are available from the density $f$. In the sequel $f$ will be called the feature density.

### 2.1 Plug-in estimators of entropy

A *truncated* kernel density estimator with uniform kernel is defined below. Our proposed weighted ensemble method applies to other types of kernels as well but we specialize to uniform kernels as it makes the derivations clearer. For integer $1 \le k \le M$, define the distance $d_k$ to be: $d_k = (k/M)^{1/d}$. Define the truncated kernel bin region for each $X \in \mathcal{S}$ to be $S_k(X) = \{Y \in \mathcal{S} : ||X - Y||_1 \le d_k/2\}$, and the volume of the truncated kernel bins to be $V_k(X) = \int_{S_k(X)} dz$. Note that when the smallest distance from $X$ to $\mathcal{S}$ is greater than $d_k$, $V_k(X) = d_k^d = k/M$. Let $\mathbf{l}_k(X)$ denotes the number of points falling in $S_k(X)$: $\mathbf{l}_k(X) = \sum_{i=1}^{M} 1_{\{\mathbf{X}_i \in S_k(X)\}}$. The truncated kernel density estimator is defined as

$$\hat{\mathbf{f}}_k(X) = \frac{\mathbf{l}_k(X)}{M V_k(X)}. \tag{2.2}$$

The plug-in estimator of the density functional is constructed using a data splitting approach as follows. The data is randomly subdivided into two parts $\{\mathbf{X}_1, \ldots, \mathbf{X}_N\}$ and $\{\mathbf{X}_{N+1}, \ldots, \mathbf{X}_{N+M}\}$ of $N$ and $M$ points respectively. In the first stage, we estimate the kernel density estimate $\hat{\mathbf{f}}_k$ at the $N$ points $\{\mathbf{X}_1, \ldots, \mathbf{X}_N\}$ using the $M$ realizations $\{\mathbf{X}_{N+1}, \ldots, \mathbf{X}_{N+M}\}$. Subsequently, we use the $N$ samples $\{\mathbf{X}_1, \ldots, \mathbf{X}_N\}$ to approximate the functional $G(f)$ and obtain the plug-in estimator:

$$\hat{\mathbf{G}}_k = \frac{1}{N} \sum_{i=1}^{N} g(\hat{\mathbf{f}}_k(\mathbf{X}_i), \mathbf{X}_i). \tag{2.3}$$

Also define a standard kernel density estimator with uniform kernel $\tilde{\mathbf{f}}_k(X)$, which is identical to $\hat{\mathbf{f}}_k(X)$ except that the volume $V_k(X)$ is always set to be $V_k(X) = k/M$. Define

$$\tilde{\mathbf{G}}_k = \frac{1}{N} \sum_{i=1}^{N} g(\tilde{\mathbf{f}}_k(\mathbf{X}_i), \mathbf{X}_i). \tag{2.4}$$

The estimator $\tilde{\mathbf{G}}_k$ is identical to the estimator of Györfi and van der Muelen [11]. Observe that the implementation of $\tilde{\mathbf{G}}_k$, unlike $\hat{\mathbf{G}}_k$, does not require knowledge about the support of the density.

#### 2.1.1 Assumptions

We make a number of technical assumptions that will allow us to obtain tight MSE convergence rates for the kernel density estimators defined in above. These assumptions are comparable to other rigorous treatments of entropy estimation. Please refer to Section II, [2] for details. $(\mathcal{A}.0)$ : Assume that the kernel bandwidth satisfies $k = k_0 M^\beta$ for any rate constant $0 < \beta < 1$, and assume that $M$, $N$ and $T$ are linearly related through the proportionality constant $\alpha_{frac}$ with: $0 < \alpha_{frac} < 1$, $M = \alpha_{frac}T$ and $N = (1 - \alpha_{frac})T$. $(\mathcal{A}.1)$ : Let the feature density $f$ be uniformly bounded away from 0 and upper bounded on the set $\mathcal{S}$, i.e., there exist constants $\epsilon_0$, $\epsilon_\infty$ such that $0 < \epsilon_0 \le f(x) \le \epsilon_\infty < \infty$ $\forall x \in \mathcal{S}$. $(\mathcal{A}.2)$: Assume that the density $f$ has continuous partial derivatives of order $d$ in the interior of the set $\mathcal{S}$, and that these derivatives are upper bounded. $(\mathcal{A}.3)$: Assume that the

function $g(f, x)$ has $\max\{\lambda, d\}$ partial derivatives w.r.t. the argument $f$, where $\lambda$ satisfies the conditions $\lambda\beta > 1$. Denote the $n$-th partial derivative of $g(f, x)$ wrt $x$ by $g^{(n)}(f, x)$. Also, let $g'(f, x) := g^{(1)}(f, x)$ and $g''(f, x) := g^{(2)}(f, x)$. $(\mathcal{A}.4)$: Assume that the absolute value of the functional $g(f, x)$ and its partial derivatives are strictly bounded away from $\infty$ in the range $\epsilon_0 < x < \epsilon_\infty$ for all $y$. $(\mathcal{A}.5)$: Let $\epsilon \in (0, 1)$ and $\delta \in (2/3, 1)$. Let $\mathcal{C}(M)$ be a positive function satisfying the condition $\mathcal{C}(M) = O(\exp(-M^{\beta(1-\delta)}))$. For some fixed $0 < \epsilon < 1$, define $p_l = (1 - \epsilon)\epsilon_0$ and $p_u = (1 + \epsilon)\epsilon_\infty$. Assume that the following four conditions are satisfied by $h(f, x) = g(f, x)$, $g^{(3)}(f, x)$ and $g^{(\lambda)}(f, x) : (i) \sup_x |h(0, x)| = G_1 < \infty$, $(ii) \sup_{f \in (p_l, p_u), x} |h(f, x)| = G_2/4 < \infty$, $(iii) \sup_{f \in (1/k, p_u), x} |h(f, x)|\mathcal{C}(M) = G_3 < \infty$, and $(iv)\mathbb{E}[\sup_{f \in (p_l, 2^d M/k), x} |h(f, x)|]\mathcal{C}(M) = G_4 < \infty$.

### 2.1.2 Analysis of MSE

Under these assumptions, we have shown the following (please see [22] for the proof) :

**Theorem 1.** *The bias of the plug-in estimators* $\hat{\mathbf{G}}_k, \tilde{\mathbf{G}}_k$ *is given by*

$$\mathbb{B}(\hat{\mathbf{G}}_k) = \sum_{i \in \mathcal{I}} c_{1,i} \left(\frac{k}{M}\right)^{i/d} + \frac{c_2}{k} + o\left(\frac{1}{k} + \frac{k}{M}\right)$$

$$\mathbb{B}(\tilde{\mathbf{G}}_k) = c_1 \left(\frac{k}{M}\right)^{1/d} + \frac{c_2}{k} + o\left(\frac{1}{k} + \frac{k}{M}\right).$$

**Theorem 2.** *The variance of the plug-in estimators* $\hat{\mathbf{G}}_k, \tilde{\mathbf{G}}_k$ *is given by*

$$\mathbb{V}(\hat{\mathbf{G}}_k) = c_4 \left(\frac{1}{N}\right) + c_5 \left(\frac{1}{M}\right) + o\left(\frac{1}{M} + \frac{1}{N}\right)$$

$$\mathbb{V}(\tilde{\mathbf{G}}_k) = c_4 \left(\frac{1}{N}\right) + c_5 \left(\frac{1}{M}\right) + o\left(\frac{1}{M} + \frac{1}{N}\right).$$

In the above expressions, $c_{1,i}$, $c_1$, $c_2$, $c_4$ and $c_5$ are constants that depend *only on* $g$, $f$ and their partial derivatives, and $\mathcal{I} = \{1, \dots, d\}$. In particular, the constants $c_{1,i}$, $c_1$, $c_2$, $c_4$ and $c_5$ are independent of $k, N$ and $M$.

### 2.1.3 Optimal MSE rate

From Theorem 1, $k \to \infty$ and $k/M \to 0$ for the estimators $\hat{\mathbf{G}}_k$ and $\tilde{\mathbf{G}}_k$ to be unbiased. Likewise from Theorem 2 $N \to \infty$ and $M \to \infty$ for the variance of the estimator to converge to 0. We can optimize the choice of bandwidth $k$, and the data splitting proportions $N/(N + M)$, $M/(N + M)$ for minimum M.S.E.

Minimizing the MSE over $k$ is equivalent to minimizing the bias over $k$. The optimal choice of $k$ is given by $k_{opt} = O(M^{1/(1+d)})$, and the bias evaluated at $k_{opt}$ is $O(M^{-1/(1+d)})$. Also observe that the MSE of $\hat{\mathbf{G}}_k$ and $\tilde{\mathbf{G}}_k$ is dominated by the squared bias $(O(M^{-2/(1+d)}))$ as contrasted to the variance $(O(1/N + 1/M))$. This implies that the asymptotic MSE rate of convergence is invariant to selected proportionality constant $\alpha_{frac}$. The optimal MSE for the estimators $\hat{\mathbf{G}}_k$ and $\tilde{\mathbf{G}}_k$ is therefore achieved for the choice of $k = O(M^{1/(1+d)})$, and is given by $O(T^{-2/(1+d)})$. In particular, observe that both $\hat{\mathbf{G}}_k$ and $\tilde{\mathbf{G}}_k$ have identical optimal rates of MSE convergence. Our goal is to reduce the estimator MSE to $O(T^{-1})$. We do so by applying the method of weighted ensembles described next in section 3.

## 3 Ensemble estimators

For a positive integer $L > d$, choose $\bar{l} = \{l_1, \dots, l_L\}$ to be a vector of distinct positive real numbers. Define the mapping $k(l) = l\sqrt{M}$ and let $\bar{k} = \{k(l); l \in \bar{l}\}$. Observe that any $k \in \bar{k}$ corresponds to the rate constant $\beta = 0.5$, and that $N = \Theta(T)$ and $M = \Theta(T)$. Define the weighted ensemble estimator

$$\hat{\mathbf{G}}_w = \sum_{l \in \bar{l}} w(l)\hat{\mathbf{G}}_{k(l)}. \tag{3.1}$$

**Theorem 3.** *There exists a weight vector $w^*$ such that*

$$\mathbb{E}[(\hat{\mathbf{G}}_{w^*} - G(f))^2] = O(1/T).$$

*This weight vector can be found by solving a convex optimization. Furthermore, this optimal weight vector does not depend on the unknown feature density $f$ or the samples $\{\mathbf{X}_1, .., \mathbf{X}_{N+M}\}$, and hence can be solved off-line.*

*Proof.* For each $i \in \mathcal{I}$, define $\gamma_w(i) = \sum_{l \in \bar{l}} w(l) l^{i/d}$. The bias of the ensemble estimator follows from Theorem 1 and is given by

$$\mathbb{B}[\hat{\mathbf{G}}_w] \quad = \quad \sum_{i \in \mathcal{I}} c_{1,i} \gamma_w(i) M^{-i/2d} + O\left(\frac{1}{\sqrt{T}}\right). \tag{3.2}$$

Denote the covariance matrix of $\{\hat{\mathbf{G}}_{k(l)}; l \in \bar{l}\}$ by $\Sigma_L$. Let $\bar{\Sigma}_L = \Sigma_L T$. Observe that by (2.5) and the Cauchy-Schwarz inequality, the entries of $\bar{\Sigma}_L$ are $O(1)$. The variance of the weighted estimator $\hat{\mathbf{G}}_w$ can then be bound as follows:

$$\mathbb{V}[\hat{\mathbf{G}}_w] \quad = \quad \mathbb{V}\left[\sum_{l \in \bar{l}} w_l \hat{\mathbf{G}}_{k(l)}\right] = w' \Sigma_L w = \frac{w' \bar{\Sigma}_L w}{T} \leq \frac{\lambda_{\max}(\bar{\Sigma}_L) \|w\|_2^2}{T}. \tag{3.3}$$

We seek a weight vector $w$ that (i) ensures that the bias of the weighted estimator is $O(T^{-1/2})$ and (ii) has low $\ell_2$ norm $\|w\|_2$ in order to limit the contribution of the variance of the weighted estimator. To this end, let $w^*$ be the solution to the convex optimization problem

$$\begin{aligned}
\underset{w}{\text{minimize}} \quad & \|w\|_2 \\
\text{subject to} \quad & \sum_{l \in \bar{l}} w(l) = 1, \\
& |\gamma_w(i)| = 0, \ i \in \mathcal{I}.
\end{aligned} \tag{3.4}$$

This problem is equivalent to minimizing $\|w\|_2$ subject to $A_0 w = b$, where $A_0$ and $b$ are defined below. Let $f_{\mathcal{IN}} : \mathcal{I} \to \{1, .., I\}$ be a bijective mapping. Let $a_0$ be the vector of ones: $[1, 1..., 1]_{1 \times L}$; and let $a_{f_{\mathcal{IN}}(i)}$, for $i \in \mathcal{I}$ be given by $a_{f_{\mathcal{IN}}(i)} = [l_1^{i/d}, .., l_L^{i/d}]$. Define $A_0 = [a_0', a_1', ..., a_I']'$, $A_1 = [a_1', ..., a_I']$ and $b = [1; 0; 0; ..; 0]_{(I+1) \times 1}$. Observe that the entries of $A_0$ and $b$ are $O(1)$, and therefore the entries of the solution $w^*$ are $O(1)$. Consequently, by (3.2), the bias $\mathbb{B}[\hat{\mathbf{G}}_{w*}] = O(1/\sqrt{T})$. Furthermore, the optimal minimum $\eta(d) := \|w^*\|_2$ is given by $\eta(d) = \sqrt{\det(A_1 A_1')/\det(A_0 A_0')}$. By (6.4), the estimator variance $\mathbb{V}[\hat{\mathbf{G}}_{w*}]$ is of order $O(\eta(d)/T)$. This concludes the proof. $\qquad \qquad \square$

While we have illustrated the weighted ensemble method only in the context of kernel estimators, this method can be applied to any general ensemble of estimators that satisfy bias and variance conditions $\mathscr{C}.1$ and $\mathscr{C}.2$ in [22].

## 4 Experiments

We illustrate the superior performance of the proposed weighted ensemble estimator for two applications: (i) estimation of the Panter-Dite rate distortion factor, and (ii) estimation of entropy to test for randomness of a random sample.

### 4.1 Panter-Dite factor estimation

For a $d$-dimensional source with underlying density $f$, the Panter-Dite distortion-rate distortion-rate function [9] for a $q$-dimensional vector quantizer with $n$ levels of quantization is given by $\delta(n) = n^{-2/q} \int f^{q/(q+2)}(x) dx$. The Panter-Dite factor corresponds to the

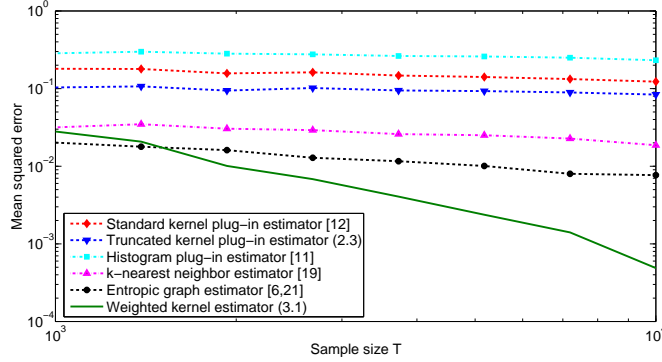

(a) Variation of MSE of Panter-Dite factor estimates as a function of sample size $T$. From the figure, we see that the proposed weighted estimator has the fastest MSE rate of convergence wrt sample size $T$.

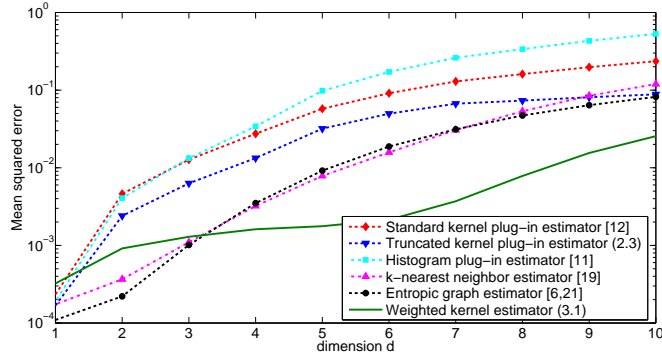

(b) Variation of MSE of Panter-Dite factor estimates as a function of dimension $d$. From the figure, we see that the MSE of the proposed weighted estimator has the slowest rate of growth with increasing dimension $d$.

Figure 1: Variation of MSE of Panter-Dite factor estimates using standard kernel plug-in estimator [12], truncated kernel plug-in estimator (2.3), histogram plug-in estimator[11], $k$-NN estimator [19], entropic graph estimator [6,21] and the weighted ensemble estimator (3.1).

functional $G(f)$ with $g(f, x) = n^{-2/q} f^{-2/(q+2)} I(f > 0) + I(f = 0)$, where $I(.)$ is the indicator function. The Panter-Dite factor is directly related to the Rényi $\alpha$-entropy, for which several other estimators have been proposed.

In our simulations we compare six different choices of functional estimators - the three estimators previously introduced: (i) the standard kernel plug-in estimator $\hat{\mathbf{G}}_k$, (ii) the boundary truncated plug-in estimator $\hat{\mathbf{G}}_k$ and (iii) the weighted estimator $\hat{\mathbf{G}}_w$ with optimal weight $w = w^*$ given by (3.4), and in addition the following popular entropy estimators: (iv) histogram plug-in estimator [10], (v) $k$-nearest neighbor ($k$-NN) entropy estimator [18] and (vi) entropic $k$-NN graph estimator [5, 20]. For both $\tilde{\mathbf{G}}_k$ and $\hat{\mathbf{G}}_k$, we select the bandwidth parameter $k$ as a function of $M$ according to the optimal proportionality $k = M^{1/(1+d)}$ and $N = M = T/2$. To illustrate the weighted estimator of the Panter-Dite factor we assume that $f$ is the $d = 6$ dimensional mixture density $f(a, b, p, d) = p f_\beta(a, b, d) + (1 - p) f_u(d)$; where $f_\beta(a, b, d)$ is a $d$-dimensional Beta density with parameters $a = 6, b = 6$, $f_u(d)$ is a $d$-dimensional uniform density and the mixing ratio $p$ is 0.8.

### 4.1.1 Variation of MSE with sample size $T$

The MSE results of these different estimators are shown in Fig. 1(a) as a function of sample size $T$. It is clear from the figure that the proposed ensemble estimator $\hat{\mathbf{G}}_w$ has significantly

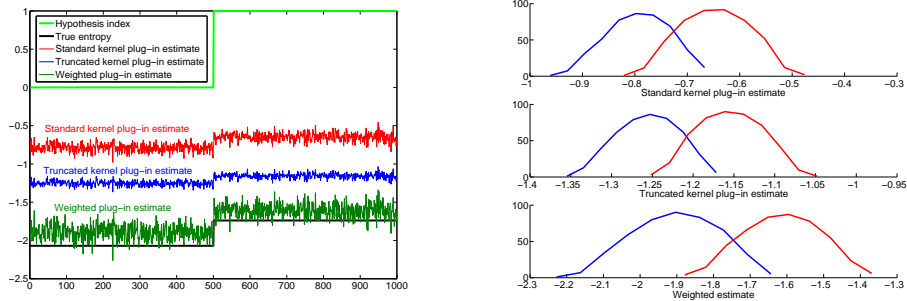

(a) Entropy estimates for random samples corresponding to hypothesis $H_0$ and $H_1$.

(b) Histogram envelopes of entropy estimates for random samples corresponding to hypothesis $H_0$ (blue) and $H_1$ (red).

Figure 2: Entropy estimates using standard kernel plug-in estimator, truncated kernel plug-in estimator and the weighted estimator, for random samples corresponding to hypothesis $H_0$ and $H_1$. The weighted estimator provided better discrimination ability by suppressing the bias, at the cost of some additional variance.

faster rate of convergence while the MSE of the rest of the estimators, including the truncated kernel plug-in estimator, have similar, slow rates of convergence. It is therefore clear that the proposed optimal ensemble averaging significantly accelerates the MSE convergence rate.

### 4.1.2 Variation of MSE with dimension $d$

The MSE results of these different estimators are shown in Fig. 1(b) as a function of dimension $d$, for fixed sample size $T = 3000$. For the standard kernel plug-in estimator and truncated kernel plug-in estimator, the MSE varies exponentially with $d$ as expected. The MSE of the histogram and $k$-NN estimators increase at a similar rate, indicating that these estimators suffer from the curse of dimensionality as well. The MSE of the weighted estimator on the other hand increases at a slower rate, which is in agreement with our theory that the MSE is $O(\eta(d)/T)$ and observing that $\eta(d)$ is an increasing function of $d$. Also observe that the MSE of the weighted estimator is significantly smaller than the MSE of the other estimators for all dimensions $d > 3$.

### 4.2 Distribution testing

In this section, Shannon differential entropy is estimated using the function $g(f, x) = -\log(f)I(f > 0) + I(f = 0)$ and used as a test statistic to test for the underlying probability distribution of a random sample. In particular, we draw 500 instances each of random samples of size $10^3$ from the probability distribution $f(a, b, p, d)$, described in Sec. 4. 1, with fixed $d = 6$, $p = 0.75$ for two sets of values of $a, b$ under the null and alternate hypothesis, $H_0 : a = a_0, b = b_0$ versus $H_1 : a = a_1, b = b_1$.

First, we fix $a_0 = b_0 = 6$ and $a_1 = b_1 = 5$. We note that the underlying density under the null hypothesis $f(6, 6, 0.75, 6)$ has greater curvature relative to $f(5, 5, 0.75, 6)$ and therefore has smaller entropy (randomness). The true entropy, and entropy estimates using $\tilde{\mathbf{G}}_k$, $\hat{\mathbf{G}}_k$ and $\hat{\mathbf{G}}_w$ for the cases corresponding to each of the 500 instances of hypothesis $H_0$ and $H_1$ are shown in Fig. 2(a). From this figure, it is apparent that the weighted estimator provides better discrimination ability by suppressing the bias, at the cost of some additional variance.

To demonstrate that the weighted estimator provides better discrimination, we plot the histogram envelope of the entropy estimates using standard kernel plug-in estimator, truncated kernel plug-in estimator and the weighted estimator for the cases corresponding to the hypothesis $H_0$ (color coded blue) and $H_1$ (color coded red) in Fig. 2(b). Furthermore, we quantitatively measure the discriminative ability of the different estimators using the deflection statistic $ds = |\mu_1 - \mu_0|/\sqrt{\sigma_0^2 + \sigma_1^2}$, where $\mu_0$ and $\sigma_0$ (respectively $\mu_1$ and $\sigma_1$) are

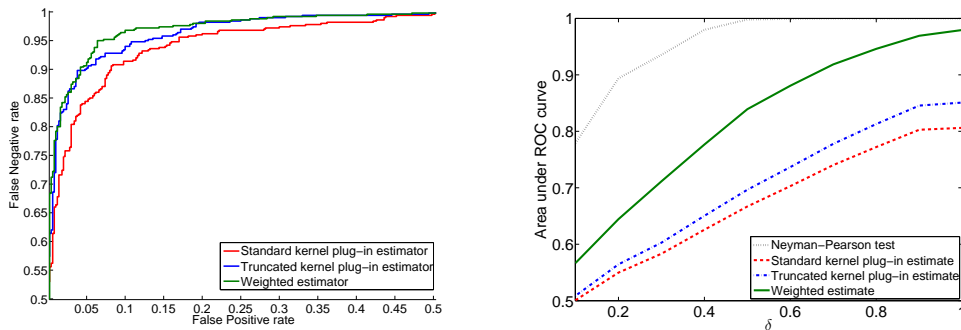

(a) ROC curves corresponding to entropy estimates obtained using standard and truncated kernel plug-in estimator and the weighted estimator. The corresponding AUC are given by 0.9271, 0.9459 and 0.9619.

(b) Variation of AUC curves vs $\delta(= a_0 - a_1, b_0 - b_1)$ corresponding to Neyman-Pearson omniscient test, entropy estimates using the standard and truncated kernel plug-in estimator and the weighted estimator.

Figure 3: Comparison of performance in terms of ROC for the distribution testing problem. The weighted estimator uniformly outperforms the individual plug-in estimators.

the sample mean and standard deviation of the entropy estimates. The deflection statistic was found to be 1.49, 1.60 and 1.89 for the standard kernel plug-in estimator, truncated kernel plug-in estimator and the weighted estimator respectively. The receiver operating curves (ROC) for this test using these three different estimators is shown in Fig. 3(a). The corresponding area under the ROC curves (AUC) are given by 0.9271, 0.9459 and 0.9619.

In our final experiment, we fix $a_0 = b_0 = 10$ and set $a_1 = b_1 = 10 - \delta$, draw 500 instances each of random samples of size $5 \times 10^3$ under the null and alternate hypothesis, and plot the AUC as $\delta$ varies from 0 to 1 in Fig. 3(b). For comparison, we also plot the AUC for the Neyman-Pearson likelihood ratio test. The Neyman-Pearson likelihood ratio test, unlike the Shannon entropy based tests, is an omniscient test that assumes knowledge of both the underlying beta-uniform mixture parametric model of the density and the parameter values $a_0$, $b_0$ and $a_1$, $b_1$ under the null and alternate hypothesis respectively. Figure 4 shows that the weighted estimator *uniformly and significantly* outperforms the individual plug-in estimators and is closest to the performance of the omniscient Neyman-Pearson likelihood test. The relatively superior performance of the Neyman-Pearson likelihood test is due to the fact that the weighted estimator is a nonparametric estimator that has marginally higher variance (proportional to $||w^*||_2^2$) compared to the underlying parametric model for which the Neyman-Pearson test statistic provides the most powerful test.

## 5 Conclusions

A novel method of weighted ensemble estimation was proposed in this paper. This method combines slowly converging individual estimators to produce a new estimator with faster MSE rate of convergence. In this paper, we applied weighted ensembles to improve the MSE of a set of uniform kernel density estimators with different kernel width parameters. We showed by theory and in simulation that that the improved ensemble estimator achieves parametric MSE convergence rate $O(T^{-1})$. The optimal weights are determined by solving a convex optimization problem which does not require training data and can be performed offline. The superior performance of the weighted ensemble entropy estimator was verified in the context of two important problems: (i) estimation of the Panter-Dite factor and (ii) non-parametric hypothesis testing.

### Acknowledgments

This work was partially supported by ARO grant W911NF-12-1-0443.

## References

[1] I. Ahmad and Pi-Erh Lin. A nonparametric estimation of the entropy for absolutely continuous distributions (corresp.). *Information Theory, IEEE Trans. on*, 22(3):372 – 375, May 1976.

[2] J. Beirlant, EJ Dudewicz, L. Györfi, and EC Van der Meulen. Nonparametric entropy estimation: An overview. *Intl. Journal of Mathematical and Statistical Sciences*, 6:17–40, 1997.

[3] L. Birge and P. Massart. Estimation of integral functions of a density. *The Annals of Statistics*, 23(1):11–29, 1995.

[4] D. Chauveau and P. Vandekerkhove. Selection of a MCMC simulation strategy via an entropy convergence criterion. *ArXiv Mathematics e-prints*, May 2006.

[5] J.A. Costa and A.O. Hero. Geodesic entropic graphs for dimension and entropy estimation in manifold learning. *Signal Processing, IEEE Transactions on*, 52(8):2210–2221, 2004.

[6] P. B. Eggermont and V. N. LaRiccia. Best asymptotic normality of the kernel density entropy estimator for smooth densities. *Information Theory, IEEE Trans. on*, 45(4):1321 –1326, May 1999.

[7] E. Giné and D.M. Mason. Uniform in bandwidth estimation of integral functionals of the density function. *Scandinavian Journal of Statistics*, 35:739761, 2008.

[8] M. Goria, N. Leonenko, V. Mergel, and P. L. Novi Inverardi. A new class of random vector entropy estimators and its applications in testing statistical hypotheses. *Nonparametric Statistics*, 2004.

[9] R. Gupta. *Quantization Strategies for Low-Power Communications*. PhD thesis, University of Michigan, Ann Arbor, 2001.

[10] L. Györfi and E. C. van der Meulen. Density-free convergence properties of various estimators of entropy. *Comput. Statist. Data Anal.*, pages 425–436, 1987.

[11] L. Györfi and E. C. van der Meulen. An entropy estimate based on a kernel density estimation. *Limit Theorems in Probability and Statistics*, pages 229–240, 1989.

[12] P. Hall and S. C. Morton. On the estimation of the entropy. *Ann. Inst. Statist. Meth.*, 45:69–88, 1993.

[13] K. Hlaváčková-Schindler, M. Paluš, M. Vejmelka, and J. Bhattacharya. Causality detection based on information-theoretic approaches in time series analysis. *Physics Reports*, 441(1):1–46, 2007.

[14] A.T. Ihler, J.W. Fisher III, and A.S. Willsky. Nonparametric estimators for online signature authentication. In *Acoustics, Speech, and Signal Processing, 2001. Proceedings.(ICASSP'01). 2001 IEEE International Conference on*, volume 6, pages 3473–3476. IEEE, 2001.

[15] H. Joe. Estimation of entropy and other functionals of a multivariate density. *Annals of the Institute of Statistical Mathematics*, 41(4):683–697, 1989.

[16] G. Lanckriet, N. Cristianini, P. Bartlett, and L. El Ghaoui. Learning the kernel matrix with semi-definite programming. *Journal of Machine Learning Research*, 5:2004, 2002.

[17] B. Laurent. Efficient estimation of integral functionals of a density. *The Annals of Statistics*, 24(2):659–681, 1996.

[18] N. Leonenko, L. Prozanto, and V. Savani. A class of Rényi information estimators for multi-dimensional densities. *Annals of Statistics*, 36:2153–2182, 2008.

[19] E. Liitiäinen, A. Lendasse, and F. Corona. On the statistical estimation of rényi entropies. In *Proceedings of IEEE/MLSP 2009 International Workshop on Machine Learning for Signal Processing, Grenoble (France)*, September 2-4 2009.

[20] D. Pal, B. Poczos, and C. Szepesvari. Estimation of Rényi entropy and mutual information based on generalized nearest-neighbor graphs. In *Proc. Advances in Neural Information Processing Systems (NIPS)*. MIT Press, 2010.

[21] Robert E. Schapire. The strength of weak learnability. *Machine Learning*, 5(2):197–227–227, June 1990.

[22] K. Sricharan and A. O. Hero, III. Ensemble estimators for multivariate entropy estimation. *ArXiv e-prints*, March 2012.

[23] C. Studholme, C. Drapaca, B. Iordanova, and V. Cardenas. Deformation-based mapping of volume change from serial brain mri in the presence of local tissue contrast change. *Medical Imaging, IEEE Transactions on*, 25(5):626–639, 2006.

[24] B. van Es. Estimating functionals related to a density by class of statistics based on spacing. *Scandinavian Journal of Statistics*, 1992.

